# Scaling MPE Inference for Constrained Continuous Markov Random Fields with Consensus Optimization

**Stephen H. Bach**
University of Maryland, College Park
College Park, MD 20742
bach@cs.umd.edu

**Matthias Broecheler**
Aurelius LLC
matthias@thinkaurelius.com

**Lise Getoor**
University of Maryland, College Park
College Park, MD 20742
getoor@cs.umd.edu

**Dianne P. O'Leary**
University of Maryland, College Park
College Park, MD 20742
oleary@cs.umd.edu

## Abstract

Probabilistic graphical models are powerful tools for analyzing constrained, continuous domains. However, finding most-probable explanations (MPEs) in these models can be computationally expensive. In this paper, we improve the scalability of MPE inference in a class of graphical models with piecewise-linear and piecewise-quadratic dependencies and linear constraints over continuous domains. We derive algorithms based on a consensus-optimization framework and demonstrate their superior performance over state of the art. We show empirically that in a large-scale voter-preference modeling problem our algorithms scale linearly in the number of dependencies and constraints.

## 1 Introduction

There is a growing need for statistical models which can capture rich dependencies in structured data. Link predication, collective classification, modeling information diffusion, entity resolution, and viral marketing are all important tasks where incorporating structural dependencies is crucial for good predictive performance. Graphical models [1] are an expressive class of statistical models to address such problems, but their applicability to large datasets is often limited by impractically expensive inference and learning algorithms.

In this paper, we focus on scaling up most-probable-explanation (MPE) inference for a particular class of graphical models called *constrained continuous Markov random fields (CCMRFs)* [2]. Like other Markov random fields (MRFs), CCMRFs define a joint distribution over a collection of random variables and capture local dependencies through potential functions. However, unlike many popular discrete MRFs which are defined over binary random variables, CCMRFs are defined over continuous random variables. They also allow their domains to be constrained. This makes CCMRFs ideally suited to reason over continuous quantities, such as similarity, affinity, or probability, *without* making assumptions about the variables' marginal distributions.[1]

MPE inference for CCMRFs is tractable under mild convexity assumptions because it can be cast as a convex numeric optimization problem, which can be solved by interior-point methods [3]. However, for large problems, interior-point methods are impractically slow because each step takes time up to cubic in the size of the problem.

We show how hinge-loss potential functions that are often used to model real world problems in CCMRFs (see, e.g., [3, 2, 4, 5, 6, 7]) can be exploited to significantly speed up the numeric optimization and therefore MPE inference. To do so, we rely on a *consensus optimization* framework [8]. Consensus optimization has recently been shown to perform well on relaxations of discrete optimization problems, like MRF MPE inference [8, 9, 10].

The contributions of this paper are as follows: First, we derive algorithms for the MPE problem in CCMRFs with piecewise-linear and piecewise-quadratic dependencies in Section 3. Next, we improve the performance of consensus optimization by deriving an algorithm that exploits opportunities for closed-form solutions to subproblems, based on the current optimization iterate, before resorting to an iterative solver when the closed-form solution is not applicable. Then, we present an experimental evaluation (Section 4) that demonstrates superior performance of our approach over a commercial interior-point method, the current state-of-the-art for CCMRF MPE inference. In a voter-preference modeling problem, our algorithms scaled linearly in the number of dependencies and constraints. In addition, compared to an exact solver, our method achieves at least $99.6\%$ of the optimal solution. Finally, we show that our improved consensus-optimization algorithm more than doubles the speed of a less sophisticated approach. To the best of our knowledge, we are the first to show results on MPE inference for any MRF variant using consensus optimization with iterative methods to solve subproblems.

## 2 Background

In this section we formally introduce the class of probabilistic graphical models for which we derive inference algorithms and present a simple running example (this is the same example used in our experiments in Section 4). We also give an overview of consensus optimization [8], the abstract framework we will use to derive our algorithms in Section 3.

### 2.1 Constrained continuous Markov random fields and the MPE problem

A constrained continuous Markov random field (CCMRF) is a probabilistic graphical model defined over continuous random variables with a constrained domain [2]. In this paper, we focus on a common subclass in which dependencies among continuous random variables are defined in terms of hinge-loss functions and linear constraints:

**Definition 1.** A hinge-loss constrained continuous Markov random field $f$ is a probability density over a finite set of $n$ random variables $\mathbf{X} = \{X_1, \ldots, X_n\}$ with domain $\mathbf{D} = [0,1]^n$. Let $\phi = \{\phi_1, \ldots, \phi_m\}$ be a finite set of $m$ continuous potential functions of the form

$$\phi_j(\mathbf{X}) = [\max\{\ell_j(\mathbf{X}), 0\}]^{p_j}$$

where $\ell_j$ is a linear function of $\mathbf{X}$ and $p_j \in \{1,2\}$. Let $C = \{C_1, \ldots, C_r\}$ be a finite set of $r$ linear constraint functions associated with two index sets denoting equality and inequality constraints, $\mathcal{E}$ and $\mathcal{I}$, which define the feasible set $\tilde{\mathbf{D}} = \{\mathbf{X} \in \mathbf{D} | C_k(\mathbf{X}) = 0, \forall k \in \mathcal{E} \text{ and } C_k(\mathbf{X}) \geq 0, \forall k \in \mathcal{I}\}$. If $\mathbf{X} \notin \tilde{\mathbf{D}}$, then $f(\mathbf{X}) = 0$. If $\mathbf{X} \in \tilde{\mathbf{D}}$, then, for a set of non-negative free parameters $\Lambda = \{\Lambda_1, \ldots, \Lambda_m\}$,

$$f(\mathbf{X}) = \frac{1}{Z(\Lambda)} \exp\left[-\sum_{j=1}^{m} \Lambda_j \phi_j(\mathbf{X})\right] \quad ; \quad Z(\Lambda) = \int_{\tilde{\mathbf{D}}} \exp\left[-\sum_{j=1}^{m} \Lambda_j \phi_j(\mathbf{X})\right] d\mathbf{X}.$$

Definition 1 is a special case of the definition of CCMRFs of Broecheler and Getoor [2]. It says that hinge-loss CCMRFs are models in which densities of assignments to variables are defined by an exponential of the negated, weighted sum of functions over those assignments, unless any constraint is violated, in which case the density is zero.

The MPE problem is to maximize $f(\mathbf{X})$ such that $\mathbf{X} \in \tilde{\mathbf{D}}$. In a hinge-loss CCMRF, the normalizing function $Z(\Lambda)$ is constant over $\mathbf{X}$ for fixed parameters and the exponential is maximized by minimizing its negated argument, so the MPE problem is

$$\arg\max_{\mathbf{X}} f(\mathbf{X}) \equiv \arg\min_{\mathbf{X} \in [0,1]^n} \sum_{j=1}^{m} \Lambda_j \phi_j(\mathbf{X}) \quad \text{s.t. } C_k(\mathbf{X}) = 0, \forall k \in \mathcal{E} \text{ and } C_k(\mathbf{X}) \geq 0, \forall k \in \mathcal{I}. \quad (1)$$

Hinge-loss CCMRFs have two main desirable properties. First, the MPE problem is convex. Second, they are expressive. Hinge-loss functions are useful for many domains. Instances of hinge-loss CCMRFs have been used previously to model many problems, including link prediction, collective classification [3, 2], prediction of opinion diffusion [4], medical decision making [5], trust analysis in social networks [6], and group detection in social networks [7].

For ease of presentation, in the rest of this paper, when we refer to CCMRFs we mean hinge-loss CCMRFs. Next, we present a motivating CCMRF, using an example from Broecheler et. al. [4].

**Example 1** (Opinion diffusion). Consider a social network $S \equiv (V, E)$ of voters in a set $V$ with relationships defined by annotated, unweighted, directed edges $(v_a, v_b)_\tau \in E$. Here, $v_a, v_b \in V$ and $\tau$ is an annotation denoting the type of relationship: `friend`, `boss`, etc. To reason about voter's opinions towards two hypothetical political parties, liberal ($L$) and conservative ($C$), we introduce two nonnegative random variables $X_{a,L}$ and $X_{a,C}$, summing to at most one, representing the strength of voter $v_a$'s preferences for each political party. We assume that $v_a$'s preference results from an *intrinsic opinion* and the influence of $v_a$'s social group. We represent the intrinsic opinion by $opinion(v_a)$, ranging from $-1$ (strongly favoring L) to 1 (strongly favoring C).

The influence of the social group is modeled by potential functions that we generically denote as $\phi$. First we penalize deviations from intrinsic opinions. If $opinion(v_a) < 0$, then $\phi \equiv [\max\{|opinion(v_a)| - X_{a,L}, 0\}]^p$, which penalizes preferences that are weaker than intrinsic opinions. Similarly, $\phi \equiv [\max\{opinion(v_a) - X_{a,C}, 0\}]^p$. when $opinion(v_a) > 0$. These hinge-loss potential functions are weighted by a fixed parameter $\Lambda_{opinion}$.

Next we penalize disagreements between voters in a social group. For each edge $(v_a, v_b)_\tau$ we introduce potential functions $\phi \equiv [\max\{X_{b,L} - X_{a,L}, 0\}]^p$ and $\phi \equiv [\max\{X_{b,C} - X_{a,C}, 0\}]^p$, penalizing preferences of $v_a$ that are not as strong as those of $v_b$. These potential functions are weighted by parameters $\Lambda_\tau$ defining the relative influence of the $\tau$ relationship. For example, we expect more influence from a close friend than from a co-worker.

We consider $p = 1$, meaning that the model has no preference between distributing the loss and accumulating it on a single potential function, and $p = 2$, meaning that that the model prefers to distribute the loss among multiple hinge-loss functions. To illustrate the choice, consider a single voter in a CCMRF with two equally-weighted potential functions $\phi_1 \equiv [\max\{0.9 - X_{a,L}, 0\}]^p$ and $\phi_2 \equiv [\max\{0.6 - X_{a,C}, 0\}]^p$. Let 0.9 and 0.6 represent the preferences of the voter's two friends. If $p = 1$, then any assignment $X_{a,L}, X_{a,C}$ with $X_{a,L} \in [0.4, 0.9]$ and $X_{a,C} = 1 - X_{a,L}$ is a MPE. However, if $p = 2$, then only the assignment $X_{a,L} = 0.65$, $X_{a,C} = 0.35$ is a MPE. We see that, all else being equal, squared potential functions "respect" the minima of individual potential functions if they cannot all be minimized. However, this useful modeling feature generally increases the computational cost. As we demonstrate in Section 4, scaling MPE inference for CCMRFs with piecewise-quadratic potential functions is one of the contributions of our work.

## 2.2 Consensus optimization

Consensus optimization is a framework that optimizes an objective by dividing it into independent subproblems and then iterating to reach a consensus on the optimum [8]. In this subsection we present an abstract consensus optimization algorithm for Problem (1), the MPE problem for CCM-RFs. In Section 3 we will derive specialized versions for different potential functions.

Given a CCMRF $(\mathbf{X}, \phi, C, \mathcal{E}, \mathcal{I}, \Lambda)$ and parameter $\rho > 0$, the algorithm first constructs a modified MPE problem in which each potential and constraint is a function of different variables. The variables are constrained to make the new and original MPE problems equivalent. We let $\mathbf{x}_j$ be a copy of the variables in $\mathbf{X}$ that are used in the potential function $\phi_j$, $j = 1, \ldots, m$ and $\mathbf{x}_{k+m}$ be a copy of those used in the constraint function $C_k$, $k = 1, \ldots, r$. We also introduce an indicator function $I_k$ for each constraint function where $I_k [C_k(\mathbf{x}_{k+m})] = 0$ if the constraint is satisfied and $\infty$ if it is not. Finally, let $\mathbf{X}_i$ be the variables in $\mathbf{X}$ that are copied in $\mathbf{x}_i$, $i = 1, \ldots, m + r$.

Consensus optimization solves the new MPE problem

$$\underset{\mathbf{x}_i \in [0,1]^{n_i}}{\arg\min} \sum_{j=1}^m \Lambda_j \phi_j(\mathbf{x}_j) + \sum_{k=1}^r I_k [C_k(\mathbf{x}_{k+m})] \quad \text{subject to} \ \mathbf{x}_i = \mathbf{X}_i \tag{2}$$

---
**Algorithm** Consensus optimization
---

**Input:** CCMRF $(\mathbf{X}, \phi, C, \mathcal{E}, \mathcal{I}, \Lambda)$, $\rho > 0$

Initialize $\mathbf{x}_j$ as a copy of the variables in $\mathbf{X}$ that appear in $\phi_j$, $j = 1, \ldots, m$
Initialize $\mathbf{x}_{k+m}$ as a copy of the variables in $\mathbf{X}$ that appear in $C_k$, $k = 1, \ldots, r$.
Initialize $\mathbf{y}_i$ at 0, $i = 1, \ldots, m + r$.
**while** not converged **do**
  **for** $i = 1, \ldots, m + r$ **do**
    $\mathbf{y}_i \leftarrow \mathbf{y}_i + \rho(\mathbf{x}_i - \mathbf{X}_i)$
  **end for**
  **for** $j = 1, \ldots, m$ **do**
    $\mathbf{x}_j \leftarrow \arg\min_{\mathbf{x}_j \in [0,1]^{n_j}} \Lambda_j \phi_j(\mathbf{x}_j) + \frac{\rho}{2}\|\mathbf{x}_j - \mathbf{X}_j + \frac{1}{\rho}\mathbf{y}_j\|_2^2$
  **end for**
  **for** $k = 1, \ldots, r$ **do**
    $\mathbf{x}_{k+m} \leftarrow \arg\min_{\mathbf{x}_{k+m} \in [0,1]^{n_{k+m}}} I_k \left[ C_k \left(\mathbf{x}_{k+m}\right)\right] + \frac{\rho}{2}\|\mathbf{x}_{k+m} - \mathbf{X}_{k+m} + \frac{1}{\rho}\mathbf{y}_{k+m}\|_2^2$
  **end for**
  Set each variable in $\mathbf{X}$ to the average of its copies
**end while**
---

where $i = 1, \ldots, m + r$ and $n_i$ is the number of components of $\mathbf{x}_i$. Inspection shows that Problems (1) and (2) are equivalent.

We use the alternating direction method of multipliers (ADMM) [11, 12, 8] to solve Problem (2). ADMM can be viewed as an approach to combining the scalability of dual decomposition and the convergence properties of augmented Lagrangian methods [8]. We outline the algorithm in the above pseudocode. At each step in the iteration, it solves $m + r$ independent optimization problems, one for each $\phi_j$ and each $C_k$. It then averages the copies of variables to get the consensus variables $\mathbf{X}$ for the next iteration. Lagrange multipliers $\mathbf{y}_i$ for each $\mathbf{x}_i$ ensure convergence. The objective is known to converge to its optimum and the iterates to approach feasibility under mild assumptions [13, 14, 8]. See Boyd et. al. [8] or this paper's supplementary material for more information. In the next section we derive algorithms with specific methods for updating each $\mathbf{x}_j$.

## 3   Solving the MPE problem with consensus optimization

We now derive algorithms to update $\mathbf{x}_j$ for each potential function $\phi_j$. At this point we drop the more complex notation and view each update as an instance of the problem

$$\arg\min_{x \in [0,1]^n} \Lambda[\max\{c^T x + c_0, 0\}]^p + (\rho/2)\|x - d\|_2^2 \tag{3}$$

where $c, d \in \mathbb{R}^n$, $c_0 \in \mathbb{R}$, $\Lambda \geq 0$, $p \in \{1, 2\}$, and $\rho > 0$. To map an update to Problem (3) for a potential function $\phi_j$ and parameter $\Lambda_j$, let $n = n_j$, $c^T x + c_0 = \ell(\mathbf{x}_j)$, $d = \mathbf{X}_j - (1/\rho)\mathbf{y}_j$, $\Lambda = \Lambda_j$, $p = p_j$, and keep $\rho$ the same.

Our first algorithm, CO-Linear, solves the MPE problem when $p = 1$ and $n \leq 2$ in each instance of Problem (3), i.e., each potential function has at most two unknowns and is piecewise-linear. We present the update in terms of the intermediate optimization problems it solves. (We use variables $\alpha$ with parenthetical superscripts to easily refer to the solutions of intermediate problems, but implementations should not treat them as separate variables.) It first finds $\alpha_1$, which is easy to do by inspection. For each component $\alpha_j^{(1)}$ of $\alpha^{(1)}$

$$\alpha_j^{(1)} = \begin{cases} 0 & \text{if } d_j < 0 \\ d_j & \text{if } 0 \leq d_j \geq 1 \\ 1 & \text{if } d_j > 1 \end{cases}$$

where $j = 1, \ldots, n$. We refer to this procedure as *clipping* the vector $d$ to the interval $[0, 1]$. In this section, when we refer to clipping to $[a, b]$, we mean an identical vector except that any component outside a bound $a$ or $b$ is changed to that bound. $\alpha_2$ is also easy to find: clip the vector $d - (\Lambda/\rho)c$ to $[0, 1]$. There are two cases when finding $\alpha^{(3)}$. If $n = 1$, clip the scalar $-c_0/c_1$ to

---

**Algorithm** Update for CO-Linear

---

**Input:** $c, d \in \mathbb{R}^n$ where $n \leq 2$, $c_0 \in \mathbb{R}$, $\Lambda \geq 0$, $\rho > 0$

**Output:** $x^\star = \arg\min_{x \in [0,1]^n} \Lambda[\max\{c^T x + c_0, 0\}] + (\rho/2)\|x - d\|_2^2$

$\alpha^{(1)} \leftarrow \arg\min_{x \in [0,1]^n} (\rho/2)\|x - d\|_2^2$    (by inspection)

**if** $c^T \alpha^{(1)} + c_0 \leq 0$ **then**
  $x^\star \leftarrow \alpha^{(1)}$
**else**
  $\alpha^{(2)} \leftarrow \arg\min_{x \in [0,1]^n} \Lambda c^T x + (\rho/2)\|x_i - d\|_2^2$    (by inspection)
  **if** $c^T \alpha^{(2)} + c_0 \geq 0$ **then**
    $x^\star \leftarrow \alpha^{(2)}$
  **else**
    $x^\star \leftarrow \alpha^{(3)} \leftarrow \arg\min_{x \in [0,1]^n \text{ s.t. } c^T x + c_0 = 0} (\rho/2)\|x - d\|_2^2$    (by substitution and inspection)
  **end if**
**end if**

---

$[0, 1]$. If $n = 2$, solve $c^T x = -c_0$ for one of the components of $x$, substitute to eliminate that component in the objective, and compute the interval $[min, max]$ on which $x \in [0, 1]^2$ when the remaining component is in $[min, max]$ and $c^T x = -c_0$. Inspect the reduced objective and clip the unconstrained minimizer to $[min, max]$. Substitute the result back into $c^T x = -c_0$ to find the other component.

To verify that the CO-Linear update is correct, first consider the case when $c^T \alpha^{(1)} + c_0 \leq 0$. Since $\alpha^{(1)}$ minimizes $(\rho/2)\|x - d\|_2^2$ and $\Lambda[\max\{c^T x + c_0, 0\}] \geq 0$, each term of the update objective is minimized at $\alpha^{(1)}$, so $x^\star = \alpha^{(1)}$. In the second case, if $c^T \alpha^{(1)} + c_0 > 0$, but $c^T \alpha^{(2)} + c_0 \geq 0$, then observe that $\alpha^{(2)}$ minimizes an objective which bounds the update objective below, but the two objectives are equal at $\alpha^{(2)}$. Therefore, $x^\star = \alpha^{(2)}$. Finally, in the third case, $c^T \alpha^{(1)} + c_0 > 0$ and $c^T \alpha^{(2)} + c_0 < 0$. We know $\exists x \in [0, 1]^n$ such that $c^T x + c_0 = 0$, so the problem can be split into two feasible problems:

$$\beta^{(1)} \equiv \arg\min_{x \in [0,1]^n \text{ s.t. } c^T x + c_0 \leq 0} (\rho/2)\|x - d\|_2^2$$

$$\beta^{(2)} \equiv \arg\min_{x \in [0,1]^n \text{ s.t. } c^T x + c_0 \geq 0} \Lambda c^T x + (\rho/2)\|x - d\|_2^2 \ .$$

Either $x^\star = \beta^{(1)}$ or $x^\star = \beta^{(2)}$ (or both). We use Lemma 4 of Martins et. al. [9] which states that given a convex, feasible optimization problem over a nonempty convex subset of $\mathbb{R}^n$ with a convex constraint, if that constraint is violated by the minimizer to a relaxed problem without that constraint over the same set, then that constraint will be active at the minimizer to the original problem. Since $c^T \alpha^{(1)} + c_0 > 0$ and $c^T \alpha^{(2)} + c_0 < 0$, we conclude that $c^T \beta^{(1)} + c_0 = 0$ and $c^T \beta^{(2)} + c_0 = 0$. Therefore $x^\star = \beta^{(1)} = \beta^{(2)} = \alpha^{(3)}$.

CO-Linear is sufficient to solve many useful and interesting models. Unfortunately, the piecewise-quadratic case ($p = 2$) is more difficult. If $n > 1$ and it cannot be established that $c^T x^\star + c_0 \leq 0$, then the approach of CO-Linear is not applicable, because minimizing $\Lambda c^T x x^T c + 2\Lambda c_0 c^T x + (\rho/2)\|x - d\|_2^2$ over $[0, 1]^n$ does not have a (known) closed-form solution in general. That motivates us to derive an algorithm for the piecewise-quadratic case that can resort to a sufficiently general iterative solver if necessary. Obviously, a naive algorithm could use an iterative method immediately if $n > 1$. However, CO-Linear still offers some insight into the problem. If clipping $d$ to $[0, 1]$ gives a vector $\alpha^{(1)}$ such that $c^T \alpha^{(1)} + c_0 \leq 0$, then again it is the minimizer.

Our second algorithm, CO-Quad, first tries to find $x^\star$ by clipping $d$ to $[0, 1]$ for any $n$. If it does not succeed and $n = 1$, then $\alpha^{(2)}$ can be found by inspection. If $n > 1$, then an iterative method is required. Note that now after concluding that $c^T x^\star + c_0 \geq 0$ we can just minimize $\Lambda c^T x x^T c + 2\Lambda c_0 c^T x + (\rho/2)\|x - d\|_2^2$ to find $x^\star$ since $\Lambda c^T x x^T c + 2\Lambda c_0 c^T x$ is symmetric about the hyperplane $c^T x + c_0 = 0$, $(\rho/2)\|x - d\|_2^2$ is minimized for some $x$ such that $c^T x + c_0 \geq 0$, and the objective is the same as the subproblem on that region.

---

**Algorithm** Update for CO-Quad

---

**Input:** $c, d \in \mathbb{R}^n$, $c_0 \in \mathbb{R}$, $\Lambda \geq 0$, $\rho > 0$

**Output:** $x^\star = \arg\min_{x \in [0,1]^n} \Lambda[\max\{c^T x + c_0, 0\}]^2 + (\rho/2)\|x - d\|_2^2$

$\alpha^{(1)} \leftarrow \arg\min_{x \in [0,1]^n}(\rho/2)\|x - d\|_2^2$    (by inspection)

**if** $c^T \alpha^{(1)} + c_0 \leq 0$ **then**
   $x^\star \leftarrow \alpha^{(1)}$
**else**
  **if** $n = 1$ **then**
    $x^\star \leftarrow \alpha^{(2)} \leftarrow \arg\min_{x \in [0,1]^n} \Lambda c^T x x^T c + 2\Lambda c_0 c^T x + (\rho/2)\|x - d\|_2^2$    (by inspection)
  **else**
    $x^\star \leftarrow \alpha^{(3)} \leftarrow \arg\min_{x \in [0,1]^n} \Lambda c^T x x^T c + 2\Lambda c_0 c^T x + (\rho/2)\|x - d\|_2^2$    (by iterative method)
  **end if**
**end if**

---

To update $\mathbf{x}_{k+m}$ for each constraint $C_k$, both CO-Linear and CO-Quad use the method proposed by Martins et. al. [9], which handles the case when $C_k(\mathbf{x}_{k+m}) = 0$ is a probability simplex. This is sufficient for the purposes of this work.

## 4 Experiments

We evaluated the scalability of CO-Linear and CO-Quad by generating social networks of varying sizes, constructing CCMRFs with them, and measuring the running time required to find a MPE. We compared our approach to the previous state-of-the-art approach for finding MPEs in CCMRFs, which uses an interior point method implemented in MOSEK, a commercial optimization package (http://www.mosek.com). Next we describe the social-network and CCMRF generation procedure, the implementations and setup, and then present the results.

### 4.1 Social-network and CCMRF generation

Our social-network generation process follows Example 1 and is based on the procedure described by Broecheler et. al. [4] to generate social networks using power-law degree distributions. Given a desired number of vertices $N$ (which the procedure matches approximately) and a list of edge types, along with parameters $\gamma$ and $\alpha$ for each type, the procedure samples in- and out-degrees for each node for each edge type from the power-law distribution $D(k) \equiv \alpha k^{-\gamma}$. Incoming and outgoing edges of the same type are then matched randomly to create edges until no more matches are possible. Vertices with no incoming or outgoing edges are removed from the network. We used six edge types with various parameters to represent relationships in social networks with different combinations of abundance and exclusivity, choosing $\gamma$ between 2 and 3, and $\alpha$ between 0 and 1, as suggested by Broecheler et. al. We then annotated each vertex with a value in $[-1, 1]$ uniformly at random to represent intrinsic opinions as described in Example 1.

We generated social networks with between 22,050 and 66,150 vertices, which induced CCMRFs with between 130,082 and 397,494 total potential functions and constraints. In all the CCMRFs, between 83% and 85% of those totals were potential functions and between 15% and 17% were constraints. For each social network, we created both a CCMRF to test CO-Linear ($p = 1$ in Definition 1) and one to test CO-Quad ($p = 2$). We chose $\Lambda_{opinion} = 0.5$ and chose $\Lambda_{\tau_1}, \ldots, \Lambda_{\tau_6}$ between 0 and 1 to model both more and less influential relationships.

### 4.2 Implementation

We implemented CO-Linear and CO-Quad in Java. We used the interior-point method in MOSEK to find $\alpha_3$ in the update for CO-Quad when necessary by passing the problem via MOSEK's Java native interface wrapper. We also compared with MOSEK's interior-point method by encoding the entire MPE problem as a linear program or a second-order cone program as appropriate, and passing the encoded problem via the Java native interface wrapper.

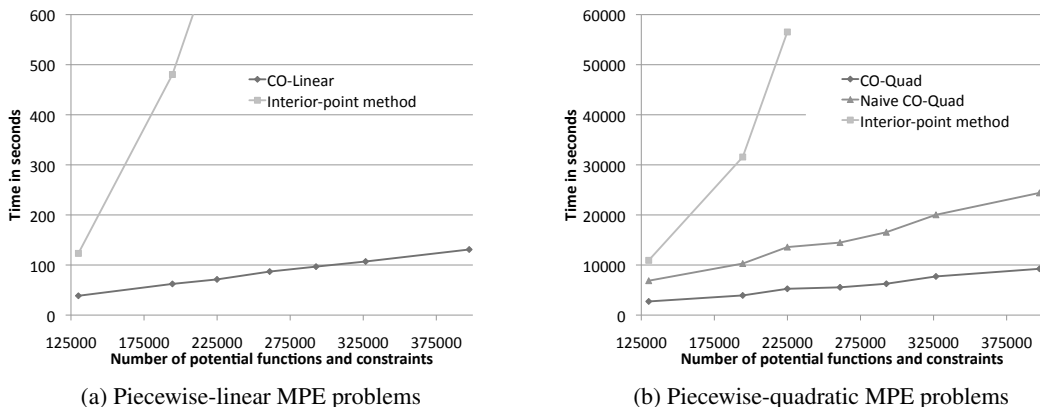

|                                  |                                     |
| (a) Piecewise-linear MPE problems | (b) Piecewise-quadratic MPE problems |

Figure 1: Average running times to find a most probable explanation (MPE) in CCMRFs.

All experiments were performed on a single machine with 2 6-core 3.06 Ghz Intel Xeon X5675 processors with 48GB of RAM. Each optimizer used a single thread. All results are averaged over 3 runs. All differences between CO-Linear and the interior-point method are significant at $p = 0.0005$. All differences between CO-Quad and the interior-point method are significant at $p = 0.005$ on problems with more than 175,000 potential functions and constraints. (The interior-point method exhibited much higher variance in running times on piecewise-quadratic problems.) All differences between CO-Quad and Naive CO-Quad are significant at $p = 0.0005$.

## 4.3 Results

We first evaluated the scalability of CO-Linear and compared with MOSEK's interior-point method. Figure 1a shows the results. The running time of the interior-point method quickly exploded as the problem size increased. Although we do not show it in the figure, the average running time on the largest problem was about 4,900 seconds (over 1 hour, 20 minutes). This demonstrates the limited scalability of the interior-point method. In contrast, CO-Linear displays excellent scalability. The average running time on the largest problem was about 130 seconds (2 minutes, 10 seconds). Further, the running time grows linearly in the number of potential functions and constraints in the CCMRF, i.e., the number of subproblems that must be solved at each iteration. The line of best fit has $R^2 = 0.99834$. Combined with Figure 1a, this shows that CO-Linear scaled linearly with increasing problem size. We emphasize that the implementation of CO-Linear is research code written in Java and the interior-point method is a commercial package running as native code. The dramatic differences in running times illustrate the superior utility of CO-Linear for these problems.

We then evaluated CO-Quad. Figure 1b shows the results (note the 2-orders-of-magnitude increase on the vertical axis between CO-Linear and CO-Quad). Again, the running time of the interior-point method quickly exploded. We could only test it on the three smallest problems, the largest of which took an average of about 56,500 seconds to solve (over 15 hours, 40 minutes). Consensus optimization again scaled linearly to the problem. The line of best fit has $R^2 = 0.9842$. To compare with the interior-point method, on the third-smallest problem, CO-Quad took an average of about 5,250 seconds (under 1 hour, 28 minutes). We also evaluated a naive variant of CO-Quad which immediately updates $\mathbf{x}_j$ via the interior-point method when there are two unknowns. As Figure 1b shows, the difference is significant. This demonstrates that CO-Quad is a further improvement on a less sophisticated approach over the previous state-of-the-art.

One of the advantages of interior-point methods is great numerical stability and accuracy, Consensus optimization, which treats both objective terms and constraints as subproblems, often returns points that are only optimal and feasible to moderate precision for non-trivially constrained problems [8]. Although this is often acceptable, we quantified the mix of infeasibility and suboptimality by repairing the infeasibility and measuring the resulting total suboptimality. We first projected the solutions returned by consensus optimization onto the feasible region, which took a negligible amount of computational time. Let $p_C$ be the value of the objective in Problem (1) at such a point and let $p_{IPM}$ be

the value of the objective at the point returned by the interior-point method. Then the relative error on that problem is $(p_C - p_{IPM})/p_{IPM}$. The relative error was consistently small. For CO-Linear, it varied between 0.2% and 0.4%, and did not trend upward as the problem size increased. For CO-Quad, when the interior-point method also returned a solution, the relative error was always less than 0.05% and also did not trend upward. This shows that consensus optimization was accurate, in addition to being dramatically faster (lower absolute time) and more scalable (smaller growth in time with problem size).

## 5 Discussion and conclusion

In this paper we advanced the state-of-the-art in solving the MPE problem for CCMRFs. With specialized algorithms, consensus optimization offers far superior scalability. In our experiments the computational cost grew linearly with the number of potential functions and constraints. This is crucially important if models are to scale to the sizes of data now available. As we build bigger models, it will be important to understand the trade-off between speed and accuracy. The well-understood theory of consensus optimization can help here. It is a major difference between our work and that of Broecheler et. al. [4], which used heuristics to solve the MPE problem by partitioning CCMRFs, fixing values of variables at the boundaries, solving relatively large subproblems with interior-point methods, and repeating with different partitions. A direction for future work is studying how to enforce desired combinations of speed and accuracy when solving MPE problems.

Such work could have a broader impact for research on solving the MPE problem for MRFs using decomposition-based approaches, which is an active area of research. Much work has studied dual decomposition for solving relaxations of discrete MPE problems [15]. Martins et. al. [9], and Meshi and Globerson [10] recently studied using consensus optimization to solve convex relaxations of the MPE problem for discrete MRFs. They solved the problem for MRFs which induced subproblems with closed-form solutions. Meshi and Globerson [10] also showed advantages of solving the dual of the relaxation and decoding the values of the discrete primal variables, but such an approach is not applicable to our work. Other recent approaches include that of Ravikumar et. al. [16], an algorithm for solving a relaxed MPE problem by solving a sequence of subproblems in a process called proximal minimization.

There are a number of remaining research problems. The first is to expand the number of unknowns in subproblems that can be solved in closed form. Another is analyzing the Karush-Kuhn-Tucker optimality conditions for the subproblems to eliminate variables when possible and solve them more efficiently. While all (hinge-loss) CCMRF subproblems could be solved with a general-purpose algorithm, such as an interior-point method, we showed that even in cases when an algorithm might have to resort to an interior-point method, exploiting opportunities for closed-form solutions greatly improved speed.

**Acknowledgments**

The authors would like to thank Neal Parikh and the anonymous reviewers for their helpful suggestions. This material is based upon work supported by the National Science Foundation under Grant No. 0937094, the Department of Energy under Grant No. DESC0002218, and the Intelligence Advanced Research Projects Activity (IARPA) via Department of Interior National Business Center (DoI/NBC) contract number D12PC00337. The U.S. Government is authorized to reproduce and distribute reprints for Governmental purposes notwithstanding any copyright annotation thereon. Disclaimer: The views and conclusions contained herein are those of the authors and should not be interpreted as necessarily representing the official policies or endorsements, either expressed or implied, of IARPA, DOI/NBA, or the U.S. Government.

**References**

[1] D. Koller and N. Friedman. *Probabilistic Graphical Models: Principles and Techniques*. The MIT Press, 2009.

[2] M. Broecheler and L. Getoor. Computing marginal distributions over continuous Markov networks for statistical relational learning. In *Advances in Neural Information Processing Systems (NIPS)*, 2010.

[3] M. Broecheler, L. Mihalkova, and L. Getoor. Probabilistic similarity logic. In *Proceedings of the 26th Conference on Uncertainty in Artificial Intelligence (UAI)*, 2010.

[4] M. Broecheler, P. Shakarian, and V. S. Subrahmanian. A scalable framework for modeling competitive diffusion in social networks. In *Proceedings of the Second International Conference on Social Computing (SocialCom)*, 2010.

[5] S. H. Bach, M. Broecheler, S. Kok, and L. Getoor. Decision-driven models with probabilistic soft logic. In *NIPS Workshop on Predictive Models in Personalized Medicine*, 2010.

[6] B. Huang, A. Kimmig, L. Getoor, and J. Golbeck. Probabilistic soft logic for trust analysis in social networks. In *International Workshop on Statistical Relational Artificial Intelligence (StaRAI)*, 2012.

[7] B. Huang, S. H. Bach, E. Norris, J. Pujara, and L. Getoor. Social group modeling with probabilistic soft logic. In *NIPS Workshop on Social Network and Social Media Analysis: Methods, Models, and Applications*, 2012.

[8] S. Boyd, N. Parikh, E. Chu, B. Peleato, and J. Eckstein. *Distributed Optimization and Statistical Learning Via the Alternating Direction Method of Multipliers*. Now Publishers, 2011.

[9] A. Martins, M. Figueiredo, P. Aguiar, N. Smith, and E. Xing. An augmented Lagrangian approach to constrained MAP inference. In *Proceedings of the 28th International Conference on Machine Learning (ICML)*, 2011.

[10] O. Meshi and A. Globerson. An alternating direction method for dual MAP LP relaxation. In *Proceedings of the 2011 European conference on machine learning and knowledge discovery in databases (ECML)*, 2011.

[11] R. Glowinski and A. Marrocco. Sur l'approximation, par éléments finis d'ordre un, et la résolution, par pénalisation-dualité, d'une classe de problèmes de Dirichlet non linéaires. *Revue française d'automatique, informatique, recherche opérationnelle*, 9(2):41–76, 1975.

[12] D. Gabay and B. Mercier. A dual algorithm for the solution of nonlinear variational problems via finite element approximation. *Computers & Mathematics with Applications*, 2(1):17–40, 1976.

[13] D. Gabay. *Applications of the method of multipliers to variational inequalities*, volume 15, chapter 9, pages 299–331. Elsevier, 1983.

[14] J. Eckstein and D. P. Bertsekas. On the Douglas-Rachford splitting method and the proximal point algorithm for maximal monotone operators. *Math. Program.*, 55(3):293–318, 1992.

[15] D. Sontag, A. Globerson, and T. Jaakkola. *Introduction to dual decomposition for inference*, chapter 8, pages 219–254. MIT Press, 2011.

[16] P. Ravikumar, A. Agarwal, and M. J. Wainwright. Message-passing for graph-structured linear programs: proximal methods and rounding schemes. *Journal of Machine Learning Research*, 11:1043–1080, 2010.

## Footnotes

[1] In contrast with Gaussian random fields where random variables are assumed to be Gaussian.
